# Kernelized Infomax Clustering

**Felix V. Agakov**
Edinburgh University
Edinburgh EH1 2QL, U.K.
felixa@inf.ed.ac.uk

**David Barber**
IDIAP Research Institute
CH-1920 Martigny Switzerland
david.barber@idiap.ch

## Abstract

We propose a simple information-theoretic approach to soft clustering based on maximizing the mutual information $I(\mathsf{x}, y)$ between the unknown cluster labels $y$ and the training patterns $\mathsf{x}$ with respect to parameters of specifically constrained encoding distributions. The constraints are chosen such that patterns are likely to be clustered similarly if they lie close to specific unknown vectors in the feature space. The method may be conveniently applied to learning the optimal affinity matrix, which corresponds to learning parameters of the kernelized encoder. The procedure does not require computations of eigenvalues of the Gram matrices, which makes it potentially attractive for clustering large data sets.

## 1   Introduction

Let $\mathsf{x} \in \mathbb{R}^{|\mathsf{x}|}$ be a visible pattern, and $y \in \{y_1, \ldots, y_{|y|}\}$ its discrete unknown cluster label. Rather than learning a density model of the observations, our goal here will be to learn a mapping $\mathsf{x} \to y$ from the observations to the latent codes (cluster labels) by optimizing a formal measure of coding efficiency. Good codes $y$ should be in some way informative about the underlying high-dimensional source vectors $\mathsf{x}$, so that the useful information contained in the sources is not lost. The fundamental measure in this context is the mutual information

$$I(\mathsf{x}, y) \overset{\text{def}}{=} H(\mathsf{x}) - H(\mathsf{x}|y) \equiv H(y) - H(y|\mathsf{x}), \tag{1}$$

which indicates the decrease in uncertainty about the pattern $\mathsf{x}$ due to the knowledge of the underlying cluster label $y$ (e.g. Cover and Thomas (1991)). Here $H(y) \equiv -\langle \log p(y) \rangle_{p(y)}$ and $H(y|\mathsf{x}) \equiv -\langle \log p(y|\mathsf{x}) \rangle_{p(\mathsf{x},y)}$ are marginal and conditional entropies respectively, and the brackets $\langle \ldots \rangle_p$ represent averages over $p$. In our case the *encoder* model is defined as

$$p(\mathsf{x}, y) \propto \sum_{m=1}^{M} \delta(\mathsf{x} - \mathsf{x}^{(m)}) p(y|\mathsf{x}), \tag{2}$$

where $\{\mathsf{x}^{(m)} | m = 1, \ldots, M\}$ is a set of training patterns.

Our goal is to maximize (1) with respect to parameters of a constrained encoding distribution $p(y|\mathsf{x})$. In contrast to most applications of the *infomax* principle

(Linsker (1988)) in stochastic channels (e.g. Brunel and Nadal (1998); Fisher and Principe (1998); Torkkola and Campbell (2000)), optimization of the objective (1) is computationally tractable since the cardinality of the code space $|y|$ (the number of clusters) will typically be low. Indeed, had the code space been high-dimensional, computation of $I(\mathsf{x}, y)$ would have required evaluation of the generally intractable entropy of the mixture $H(\mathsf{y})$, and approximations would have needed to be considered (e.g. Barber and Agakov (2003); Agakov and Barber (2006)).

Maximization of the mutual information with respect to parameters of the encoder model effectively defines a *discriminative unsupervised* optimization framework, where the model is parameterized similarly to a conditionally trained classifier, but where the cluster allocations are generally unknown. Training such models $p(y|\mathsf{x})$ by maximizing the likelihood $p(\mathsf{x})$ would be meaningless, as the cluster variables would marginalize out, which motivates also our information theoretic approach. In this way we may extract soft cluster allocations directly from the training set, with no additional information about class labels, relevance patterns, etc. required. This is an important difference from other clustering techniques making a recourse to information theory, which consider different channels and generally require additional information about relevance or irrelevance variables (*cf* Tishby et al. (1999); Chechik and Tishby (2002); Dhillon and Guan (2003)).

Our infomax approach is in contrast with probabilistic methods based on likelihood maximization. There the task of finding an optimal cluster allocation $y$ for an observed pattern $\mathsf{x}$ may be viewed as an inference problem in *generative* models $y \to \mathsf{x}$, where the probability of the data $p(\mathsf{x}) = \sum_y p(y)p(\mathsf{x}|y)$ is defined as a mixture of $|y|$ processes. The key idea of fitting such models to data is to find a constrained probability distribution $p(\mathsf{x})$ which would be likely to generate the visible patterns $\{\mathsf{x}^{(1)}, \dots, \mathsf{x}^{(M)}\}$ (this is commonly achieved by maximizing the marginal likelihood for deterministic parameters of the constrained distribution). The unknown clusters $y$ corresponding to each pattern $\mathsf{x}$ may then be assigned according to the posterior $p(y|\mathsf{x}) \propto p(y)p(\mathsf{x}|y)$. Such generative approaches are well-known but suffer from the constraint that $p(\mathsf{x}|y)$ is a correctly normalised distribution in $\mathsf{x}$. In high dimensions $|\mathsf{x}|$ this restricts the class of generative distributions usually to (mixtures of) Gaussians whose mean is dependent (in a linear or non-linear way) on the latent cluster $y$. Typically data will lie on low dimensional curved manifolds embedded in the high dimensional $\mathsf{x}$-space. If we are restricted to using mixtures of Gaussians to model this curved manifold, typically a very large number of mixture components will be required. No such restrictions apply in the infomax case so that the mappings $p(y|\mathsf{x})$ may be very complex, subject only to sensible clustering constraints.

## 2    Clustering in Nonlinear Encoder Models

Arguably, there are at least two requirements which a meaningful cluster allocation procedure should satisfy. Firstly, clusters should be, in some sense, locally smooth. For example, each pair of source vectors should have a high probability of being assigned to the same cluster if the vectors satisfy specific geometric constraints. Secondly, we may wish to avoid assigning unique cluster labels to outliers (or other constrained regions in the data space), so that under-represented regions in the data space are not over-represented in the code space. Note that degenerate cluster allocations are generally suboptimal under the objective (1), as they would lead to a reduction in the marginal entropy $H(y)$. On the other hand, it is intuitive that maximization of the mutual information $I(\mathsf{x}, y)$ favors hard assignments of cluster labels to equiprobable data regions, as this would result in the growth in $H(y)$ and reduction in $H(y|\mathsf{x})$.

## 2.1 Learning Optimal Parameters

Local smoothness and "softness" of the clusters may be enforced by imposing appropriate constraints on $p(y|\mathsf{x})$. A simple choice of the encoder is

$$p(y_j|\mathsf{x}^{(i)}) \propto \exp\{-\|\mathsf{x}^{(i)} - \mathsf{w}_j\|^2/s_j + b_j\}, \qquad (3)$$

where the cluster centers $\mathsf{w}_j \in \mathbb{R}^{|\mathsf{x}|}$, the dispersions $s_j$, and the biases $b_j$ are the encoder parameters to be learned. Clearly, under the encoding distribution (3) patterns $\mathsf{x}$ lying close to specific centers $\mathsf{w}_j$ in the *data* space will tend to be clustered similarly. In principle, we could consider other choices of $p(y|\mathsf{x})$; however (3) will prove to be particularly convenient for the kernelized extensions.

Learning the optimal cluster allocations corresponds to maximizing (1) with respect to the encoder parameters (3). The gradients are given by

$$\frac{\partial I(\mathsf{x}, y)}{\partial \mathsf{w}_j} = \frac{1}{M} \sum_{m=1}^{M} p(y_j|\mathsf{x}^{(m)}) \frac{(\mathsf{x}^{(m)} - \mathsf{w}_j)}{s_j} \alpha_j^{(m)} \qquad (4)$$

$$\frac{\partial I(\mathsf{x}, y)}{\partial s_j} = \frac{1}{M} \sum_{m=1}^{M} p(y_j|\mathsf{x}^{(m)}) \frac{\|\mathsf{x}^{(m)} - \mathsf{w}_j\|^2}{2s_j^2} \alpha_j^{(m)}. \qquad (5)$$

Analogously, we get $\partial I(\mathsf{x}, y)/\partial b_j = \sum_{m=1}^{M} p(y_j|\mathsf{x}^{(m)}) \alpha_j^{(m)}/M$.

Expressions (4) and (5) have the form of the weighted EM updates for isotropic Gaussian mixtures, with the weighting coefficients $\alpha_j^{(m)}$ defined as

$$\alpha_j^{(m)} \stackrel{\text{def}}{=} \alpha_j(\mathsf{x}^{(m)}) \stackrel{\text{def}}{=} \log \frac{p(y_j|\mathsf{x}^{(m)})}{p(y_j)} - KL\left(p(y|\mathsf{x}^{(m)})\|\langle p(y|\mathsf{x})\rangle_{\tilde{p}(\mathsf{x})}\right), \qquad (6)$$

where $KL$ defines the Kullback-Leibler divergence (e.g. Cover and Thomas (1991)), and $\tilde{p}(\mathsf{x}) \propto \sum_m \delta(\mathsf{x} - \mathsf{x}^{(m)})$ is the empirical distribution. Clearly, if $\alpha_j^{(m)}$ is kept fixed for all $m = 1, \ldots, M$ and $j = 1, \ldots, |y|$, the gradients (4) are identical to those obtained by maximizing the log-likelihood of a Gaussian mixture model (up to irrelevant constant pre-factors). Generally, however, the coefficients $\alpha_j^{(m)}$ will be functions of $\mathsf{w}_l, s_l$, and $b_l$ for all cluster labels $l = 1, \ldots, |y|$.

In practice, we may impose a simple construction ensuring that $s_j > 0$, for example by assuming that $s_j = \exp\{\tilde{s}_j\}$ where $\tilde{s}_j \in \mathbb{R}$. For this case, we may re-express the gradients for the variances as $\partial I(\mathsf{x}, y)/\partial \tilde{s}_j = s_j \partial I(\mathsf{x}, y)/\partial s_j$. Expressions (4) and (5) may then be used to perform gradient ascent on $I(\mathsf{x}, y)$ for $\mathsf{w}_j$, $\tilde{s}_j$, and $b_j$, where $j = 1, \ldots, |y|$. After training, the optimal cluster allocations may be assigned according to the encoding distribution $p(y|\mathsf{x})$.

## 2.2 Infomax Clustering with Kernelized Encoder Models

We now extend (3) by considering a kernelized parameterization of a nonlinear encoder. Let us assume that the source patterns $\mathsf{x}^{(i)}$, $\mathsf{x}^{(j)}$ have a high probability of being assigned to the same cluster if they lie close to a specific cluster center in some *feature* space. One choice of the encoder distribution for this case is

$$p(y_j|\mathsf{x}^{(i)}) \propto \exp\{-\|\boldsymbol{\phi}(\mathsf{x}^{(i)}) - \mathsf{w}_j\|^2/s_j + b_j\}, \qquad (7)$$

where $\boldsymbol{\phi}(\mathsf{x}^{(i)}) \in \mathbb{R}^{|\phi|}$ is the feature vector corresponding to the source pattern $\mathsf{x}^{(i)}$, and $\mathsf{w}_j \in \mathbb{R}^{|\phi|}$ is the (unknown) cluster center in the feature space. The feature space may be very high- or even infinite-dimensional.

Since each cluster center $\mathsf{w}_i \in \mathbb{R}^{|\phi|}$ lives in the same space as the projected sources $\phi(\mathsf{x}^{(i)})$, it is representable in the basis of the projections as

$$\mathsf{w}_j = \sum_{m=1}^{M} \alpha_{mj} \phi(\mathsf{x}^{(m)}) + \mathsf{w}_j^{\perp}, \tag{8}$$

where $\tilde{\mathsf{w}}_i^{\perp} \in \mathbb{R}^{|\phi|}$ is orthogonal to the span of $\phi(\mathsf{x}_1), \ldots, \phi(\mathsf{x}_M)$, and $\{\alpha_{mj}\}$ is a set of coefficients (here $j$ and $m$ index $|y|$ codes and $M$ patterns respectively). Then we may transform the encoder distribution (7) to

$$
\begin{aligned}
p(y_j|\mathsf{x}^{(m)}) &\propto \exp\left\{-\left(K_{mm} - 2\mathsf{k}^T(\mathsf{x}^{(m)})\mathsf{a}_j + \mathsf{a}_j^T \mathsf{K}\mathsf{a}_j + c_j\right)/s_j\right\} \\
&\stackrel{\text{def}}{=} \exp\{-f_j(\mathsf{x}^{(m)})\},
\end{aligned} \tag{9}
$$

where $\mathsf{k}(\mathsf{x}^{(m)})$ corresponds to the $m^{th}$ column (or row) of the Gram matrix $\mathsf{K} \stackrel{\text{def}}{=} \{K_{ij}\} \stackrel{\text{def}}{=} \{\phi(\mathsf{x}^{(i)})^T \phi(\mathsf{x}^{(j)})\} \in \mathbb{R}^{M \times M}$, $\mathsf{a}_j \in \mathbb{R}^M$ is the $j^{th}$ column of the matrix of the coefficients $\mathsf{A} \stackrel{\text{def}}{=} \{a_{mj}\} \in \mathbb{R}^{M \times |y|}$, and $c_j = (\mathsf{w}_j^{\perp})^T \mathsf{w}_j^{\perp} - s_j b_j$. Without loss of generality, we may assume that $\mathsf{c} = \{c_j\} \in \mathbb{R}^{|y|}$ is a free unconstrained parameter. Additionally, we will ensure positivity of the dispersions $s_j$ by considering a construction constraint $s_j = \exp\{\tilde{s}_j\}$, where $\tilde{s}_j \in \mathbb{R}$.

**Learning Optimal Parameters**

First we will assume that the Gram matrix $\mathsf{K} \in \mathbb{R}^{M \times M}$ is *fixed* and *known* (which effectively corresponds to considering a fixed affinity matrix, see e.g. Dhillon et al. (2004)). Objective (1) should be optimized with respect to the log-dispersions $\tilde{s}_j \equiv \log(s_j)$, biases $c_j$, and coordinates $\mathsf{A} \in \mathbb{R}^{M \times |y|}$ in the space spanned by the feature vectors $\{\phi(\mathsf{x}^{(i)})|i = 1, \ldots, M\}$. From (9) we get

$$\frac{\partial I(\mathsf{x}, y)}{\partial \mathsf{a}_j} = \frac{1}{s_j} \langle p(y_j|\mathsf{x}) (\mathsf{k}(\mathsf{x}) - \mathsf{K}\mathsf{a}_j) \alpha_j(\mathsf{x}) \rangle_{\tilde{p}(\mathsf{x})} \in \mathbb{R}^M, \tag{10}$$

$$\frac{\partial I(\mathsf{x}, y)}{\partial \tilde{s}_j} = \frac{1}{2s_j} \langle p(y_j|\mathsf{x}) f_j(\mathsf{x}) \alpha_j(\mathsf{x}) \rangle_{\tilde{p}(\mathsf{x})}, \tag{11}$$

where $\tilde{p}(\mathsf{x}) \propto \sum_{m-1}^{M} \delta(\mathsf{x} - \mathsf{x}^{(m)})$ is the empirical distribution. Analogously, we obtain

$$\partial I(\mathsf{x}, y)/\partial c_j = \langle \alpha_j(\mathsf{x}) \rangle_{\tilde{p}(\mathsf{x})}, \tag{12}$$

where the coefficients $\alpha_j(\mathsf{x})$ are given by (6). For a known Gram matrix $\mathsf{K} \in \mathbb{R}^{M \times M}$, the gradients $\partial I/\partial \mathsf{a}_j$, $\partial I/\partial \tilde{s}_j$, and $\partial I/\partial c_j$ given by expressions (10) – (12) may be used in numerical optimization for the model parameters. Note that the matrix multiplication in (10) is performed once for each $\mathsf{a}_j$, so that the complexity of computing the gradient is $\sim O(M^2|y|)$ per iteration. We also note that one could potentially optimize (1) by applying the iterative Arimoto-Blahut algorithm for maximizing the channel capacity (see e.g. Cover and Thomas (1991)). However, for any given *constrained* encoder it is generally difficult to derive closed-form updates for the parameters of $p(y|\mathsf{x})$, which motivates a numerical optimization.

**Learning Optimal Kernels**

Since we presume that explicit computations in $\mathbb{R}^{|\phi|}$ are expensive, we cannot compute the Gram matrix by trivially applying its definition $\mathsf{K} = \{\phi(\mathsf{x}_i)^T \phi(\mathsf{x}_j)\}$. Instead, we may interpret scalar products in feature spaces as *kernel functions*

$$\phi(\mathsf{x}^{(i)})^T \phi(\mathsf{x}^{(j)}) = \mathcal{K}_{\Theta}(\mathsf{x}^{(i)}, \mathsf{x}^{(j)}; \Theta), \quad \forall \mathsf{x}^{(i)}, \mathsf{x}^{(j)} \in \mathcal{R}_{\mathsf{x}}, \tag{13}$$

where $\mathcal{K}_\Theta : \mathcal{R}_x \times \mathcal{R}_x \to \mathbb{R}$ satisfies Mercer's kernel properties (e.g. Scholkopf and Smola (2002)). We may now apply our *unsupervised* framework to implicitly learn the optimal nonlinear features by optimizing $I(x, y)$ with respect to the parameters $\Theta$ of the kernel function $\mathcal{K}_\Theta$. After some algebraic manipulations, we get

$$M\frac{\partial I(x, y)}{\partial \Theta} = \sum_{m=1}^{M} KL(p(y|x^{(m)})\|p(y)) \sum_{k=1}^{|y|} \frac{\partial f_k(x^{(m)})}{\partial \Theta} p(y_k|x^{(m)})$$
$$- \sum_{m=1}^{M} \sum_{j=1}^{|y|} \frac{\partial f_j(x^{(m)})}{\partial \Theta} p(y_j|x^{(m)}) \log \frac{p(y_j|x^{(m)})}{p(y_j)} \quad (14)$$

where $f_k(x^{(m)})$ is given by (9). The computational complexity of computing the updates for $\Theta$ is $O(M|y|^2)$, where $M$ is the number of training patterns and $|y|$ is the number of clusters (which is assumed to be small). Note that in contrast to spectral methods (see e.g. Shi and Malik (2000), Ng et al. (2001)) neither the objective (1) nor its gradients require inversion of the Gram matrix $K \in \mathbb{R}^{M \times M}$ or computations of its eigenvalue decomposition.

In the special case of the radial basis function (RBF) kernels

$$\mathcal{K}_\beta(x^{(i)}, x^{(j)}) = \exp\{-\beta\|x^{(i)} - x^{(j)}\|^2\}, \quad (15)$$

the gradients of the encoder potentials are simply given by

$$\frac{\partial f_j(x^{(m)})}{\partial \beta} = \frac{1}{s_j} \left( a_j^T \tilde{K} a_j - 2\tilde{k}^T(x^{(m)}) a_j \right), \quad (16)$$

where $\tilde{K} \stackrel{\text{def}}{=} \{\tilde{K}_{ij}\} \stackrel{\text{def}}{=} K(x^{(i)}, x^{(j)})(1 - \delta(x^{(i)} - x^{(j)}))$, and $\delta$ is the Kronecker delta. By substituting (16) into the general expression (14), we obtain the gradient of the mutual information with respect to the RBF kernel parameters.

## 3 Demonstrations

We have empirically compared our kernelized information-theoretic clustering approach with Gaussian mixture, k-means, feature-space k-means, non-kernelized information-theoretic clustering (see Section 2.1), and a multi-class spectral clustering method optimizing the *normalized cuts*. We illustrate the methods on datasets that are particularly easy to visualize. Figure 1 shows a typical application of the methods to the spiral data, where $x_1(t) = t\cos(t)/4$, $x_2(t) = t\sin(t)/4$ correspond to different coordinates of $x \in \mathbb{R}^{|x|}, |x| = 2$, and $t \in [0, 3.(3)\pi]$. The kernel parameters $\beta$ of the RBF-kernelized encoding distribution were initialized at $\beta_0 = 2.5$ and learned according to (16). The initial settings of the coefficients $A \in \mathbb{R}^{M \times |y|}$ in the feature space were sampled from $\mathcal{N}_{A_{ij}}(0, 0.1)$. The log-variances $\tilde{s}_1, \ldots, \tilde{s}_{|y|}$ were initialized at zeros. The encoder parameters $A$ and $\{\tilde{s}_j | j = 1, \ldots, |y|\}$ (along with the RBF kernel parameter $\beta$) were optimized by applying the scaled conjugate gradients. We found that Gaussian mixtures trained by maximizing the likelihood usually resulted in highly stochastic cluster allocations; additionally, they led to a large variation in cluster sizes. The Gaussian mixtures were initialized using k-means – other choices usually led to worse performance. We also see that the k-means effectively breaks, as the similarly clustered points lie close to each other in $\mathbb{R}^2$ (according to the $L_2$-norm), but the allocated clusters are *not* locally smooth in $t$. On the other hand, our method with the RBF-kernelized encoders typically led to locally smooth cluster allocations.

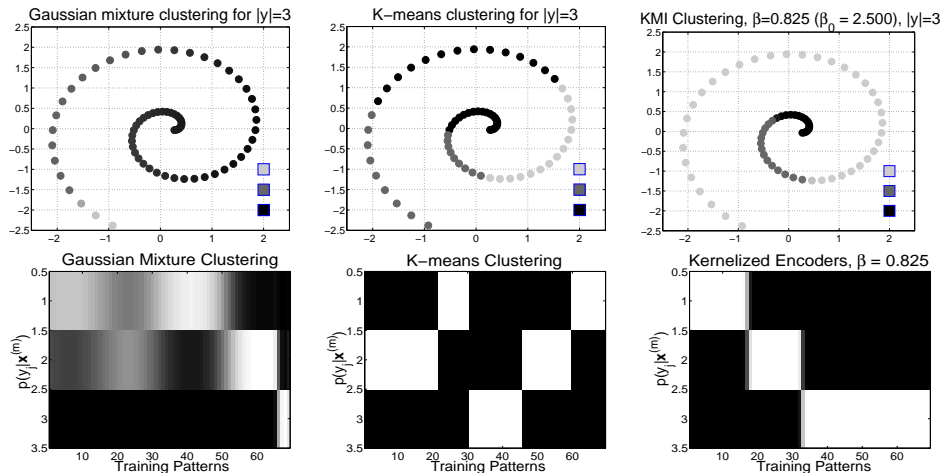

Figure 1: Cluster allocations (*top*) and the corresponding responsibilities (*bottom*) $p(y_j|\mathbf{x}^{(m)})$ for $|\mathbf{x}| = 2$, $|y| = 3$, $M = 70$ (the patterns are sorted to indicate local smoothness in the phase parameter). *Left:* Gaussian mixtures; *middle:* K-means; *right:* information-maximization for the (RBF-)kernelized encoder (the learned parameter $\beta \approx 0.825$). Light, medium, and dark-gray squares show the cluster colors corresponding to deterministic cluster allocations. The color intensity of each training point $\mathbf{x}^{(m)}$ is the average of the pure cluster intensities, weighted by the responsibilities $p(y_j|\mathbf{x}^{(m)})$. Nearly indistinguishable dark colors of the Gaussian mixture clustering indicate soft cluster assignments.

Figure 2 shows typical results for spatially translated letters with $|\mathbf{x}| = 2$, $M = 150$, and $|y| = 2$ (or $|y| = 3$), where we compare Gaussian mixture, feature-space k-means, the spectral method of Ng et al. (2001), and our information-theoretic clustering method. The initializations followed the same procedure as the previous experiment. The results produced by our kernelized infomax method were generally stable under different initializations, provided that $\beta_0$ was not too large or too small. In contrast to Gaussian mixture, spectral, and feature-space k-means clustering, the clusters produced by kernelized infomax for the cases considered are arguably more anthropomorphically appealing. Note that feature-space k-means, as well as the spectral method, presume that the kernel matrix $\mathsf{K} \in \mathbb{R}^{M \times M}$ is fixed and known (in the latter case, the Gram matrix defines the edge weights of the graph). For illustration purposes, we show the results for the fixed Gram matrices with kernel parameters $\beta$ set to the initial values $\beta_0 = 1$ or the learned values $\beta \approx 0.604$ of the kernelized infomax method for $|y| = 2$. One may potentially improve the performance of these methods by running the algorithms several times (with different kernel parameters $\beta$), and choosing $\beta$ which results in tightest clusters (Ng et al. (2001)). We were indeed able to apply the spectral method to obtain clusters for *TA* and *T* (for $\beta \approx 1.1$). While being useful in some situations, the procedure generally requires multiple runs. In contrast, the kernelized infomax method typically resulted in meaningful cluster allocations (*TT* and *A*) after a single run of the algorithm (see Figure 2 (*c*)), with the results qualitatively consistent under a variety of initializations.

Additionally, we note that in situations when we used simpler encoder models (see expression (3)) or did not adapt parameters of the kernel functions, the extracted clusters were often more intuitive than those produced by rival methods, but inferior

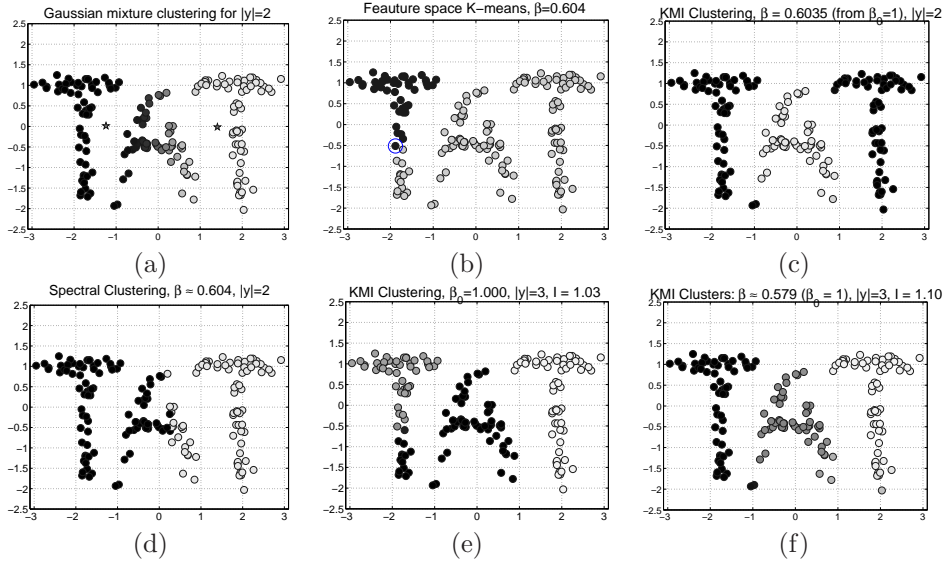

Figure 2: Learning cluster allocations for $|y| = 2$ and $|y| = 3$. Where appropriate, the stars show the cluster centers. *(a)* two-component Gaussian mixture trained by the EM algorithm; *(b)* feature-space k-means with $\beta = 1.0$ and $\beta \approx 0.604$ (the only pattern clustered differently (under identical initializations) is shown by ⊚); *(c)* kernelized infomax clustering for $|y| = 2$ (the inverse variance $\beta$ of the RBF kernel varied from $\beta_0 = 1$ (at the initialization) to $\beta \approx 0.604$ after convergence); *(d)* spectral clustering for $|y| = 2$ and $\beta \approx 0.604$; *(e)* kernelized infomax clustering for $|y| = 3$ with a *fixed* Gram matrix; *(f)* kernelized infomax clustering for $|y| = 3$ started at $\beta_0 = 1$ and reaching $\beta \approx 0.579$ after convergence.

to the ones produced by (7) with the optimal learned $\beta$. Our results suggest that by learning kernel parameters we may often obtain higher values of the objective $I(\mathsf{x}, y)$, as well as more appealing cluster labeling (e.g. for the examples shown on Figure 2 *(e)*, *(f)* we get $I(\mathsf{x}, y) \approx 1.03$ and $I(\mathsf{x}, y) \approx 1.10$ respectively). Undoubtedly, a careful choice of the kernel function could potentially lead to an even better visualization of the locally smooth, non-degenerate structure.

## 4 Discussion

The proposed information-theoretic clustering framework is fundamentally different from the generative latent variable clustering approaches. Instead of explicitly parameterizing the data-generating process, we impose constraints on the *encoder* distributions, transforming the clustering problem to learning optimal discrete encodings of the unlabeled data. Many possible parameterizations of such distributions may potentially be considered. Here we discussed one such choice, which implicitly utilizes projections of the data to high-dimensional feature spaces.

Our method suggests a formal information-theoretic procedure for learning optimal cluster allocations. One potential disadvantage of the method is a potentially large number of local optima; however, our empirical results suggest that the method is stable under different initializations, provided that the initial variances are sufficiently large. Moreover, the results suggest that in the cases considered the method

favorably compares with the common generative clustering techniques, k-means, feature-space k-means, and the variants of the method which do not use nonlinearities or do not learn parameters of kernel functions.

A number of interesting interpretations of clustering approaches in feature spaces are possible. Recently, it has been shown (Bach and Jordan (2003); Dhillon et al. (2004)) that spectral clustering methods optimizing normalized cuts (Shi and Malik (2000); Ng et al. (2001)) may be viewed as a form of weighted feature-space k-means, for a specific fixed similarity matrix. We are currently relating our method to the common spectral clustering approaches and a form of annealed weighted feature-space k-means. We stress, however, that our information-maximizing framework suggests a principled way of learning optimal similarity matrices by adapting parameters of the kernel functions. Additionally, our method does not require computations of eigenvalues of the similarity matrix, which may be particularly beneficial for large datasets. Finally, we expect that the proper information-theoretic interpretation of the encoder framework may facilitate extensions of the information-theoretic clustering method to richer families of encoder distributions.

## References

Agakov, F. V. and Barber, D. (2006). Auxiliary Variational Information Maximization for Dimensionality Reduction. In *Proceedings of the PASCAL Workshop on Subspace, Latent Structure and Feature Selection Techniques*. Springer. To appear.

Bach, F. R. and Jordan, M. I. (2003). Learning spectral clustering. In *NIPS*. MIT Press.

Barber, D. and Agakov, F. V. (2003). The IM Algorithm: A Variational Approach to Information Maximization. In *NIPS*. MIT Press.

Brunel, N. and Nadal, J.-P. (1998). Mutual Information, Fisher Information and Population Coding. *Neural Computation*, 10:1731–1757.

Chechik, G. and Tishby, N. (2002). Extracting relevant structures with side information. In *NIPS*, volume 15. MIT Press.

Cover, T. M. and Thomas, J. A. (1991). *Elements of Information Theory*. Wiley, NY.

Dhillon, I. S. and Guan, Y. (2003). Information Theoretic Clustering of Sparse Co-Occurrence Data. In *Proceedings of the $3^{rd}$ IEEE International Conf. on Data Mining*.

Dhillon, I. S., Guan, Y., and Kulis, B. (2004). Kernel k-means, Spectral Clustering and Normalized Cuts. In *KDD*. ACM.

Fisher, J. W. and Principe, J. C. (1998). A methodology for information theoretic feature extraction. In *Proc. of the IEEE International Joint Conference on Neural Networks*.

Linsker, R. (1988). Towards an Organizing Principle for a Layered Perceptual Network. In *Advances in Neural Information Processing Systems*. American Institute of Physics.

Ng, A. Y., Jordan, M., and Weiss, Y. (2001). On spectral clustering: Analysis and an algorithm. In *NIPS*, volume 14. MIT Press.

Scholkopf, B. and Smola, A. (2002). *Learning with Kernels*. MIT Press.

Shi, J. and Malik, J. (2000). Normalized Cuts and Image Segmentation. *IEEE Transactions on Pattern Analysis and Machine Intelligence*, 22(8):888–905.

Tishby, N., Pereira, F. C., and Bialek, W. (1999). The information bottleneck method. In *Proceedings of the 37-th Annual Allerton Conference on Communication, Control and Computing*. Kluwer Academic Publishers.

Torkkola, K. and Campbell, W. M. (2000). Mutual Information in Learning Feature Transformations. In *ICML*. Morgan Kaufmann.
